# Skill characterization based on betweenness

**Özgür Şimşek**[*]
**Andrew G. Barto**
Department of Computer Science
University of Massachusetts
Amherst, MA 01003
{ozgur|barto}@cs.umass.edu

## Abstract

We present a characterization of a useful class of skills based on a graphical representation of an agent's interaction with its environment. Our characterization uses *betweenness*, a measure of centrality on graphs. It captures and generalizes (at least intuitively) the *bottleneck* concept, which has inspired many of the existing skill-discovery algorithms. Our characterization may be used directly to form a set of skills suitable for a given task. More importantly, it serves as a useful guide for developing incremental skill-discovery algorithms that do not rely on knowing or representing the interaction graph in its entirety.

## 1 Introduction

The broad problem we consider is how to equip artificial agents with the ability to form useful high-level behaviors, or *skills*, from available primitives. For example, for a robot performing tasks that require manipulating objects, grasping is a useful skill that employs lower-level sensory and motor primitives. In approaching this problem, we distinguish between two related questions: What constitutes a useful skill? And, how can an agent identify such skills autonomously? Here, we address the former question with the objective of guiding research on the latter.

Our main contribution is a characterization of a useful class of skills based on a graphical representation of the agent's interaction with its environment. Specifically, we use *betweenness*, a measure of centrality on graphs [1, 2], to define a set of skills that allows efficient navigation on the interaction graph. In the game of Tic-Tac-Toe, these skills translate into setting up a fork, creating an opportunity to win the game. In the Towers of Hanoi puzzle, they include clearing the stack above the largest disk and clearing one peg entirely, making it possible to move the largest disk.

Our characterization may be used directly to form a set of skills suitable for a given task if the interaction graph is readily available. More importantly, this characterization is a useful guide for developing low-cost, incremental algorithms for skill discovery that do not rely on complete representation of the interaction graph. We present one such algorithm here and perform preliminary analysis.

Our characterization captures and generalizes (at least intuitively) the *bottleneck* concept, which has inspired many of the existing skill-discovery algorithms [3, 4, 5, 6, 7, 8, 9]. Bottlenecks have been described as *regions that the agent tends to visit frequently on successful trajectories but not on unsuccessful ones* [3], *border states of strongly connected areas* [6], and *states that allow transitions to a different part of the environment* [7]. The canonical example is a doorway connecting two rooms. We hope that our explicit and concrete description of what makes a useful skill will lead to further development of these existing algorithms and inspire alternative methods.

---

[*]Now at the Max Planck Institute for Human Development, Center for Adaptive Behavior and Cognition, Berlin, Germany.

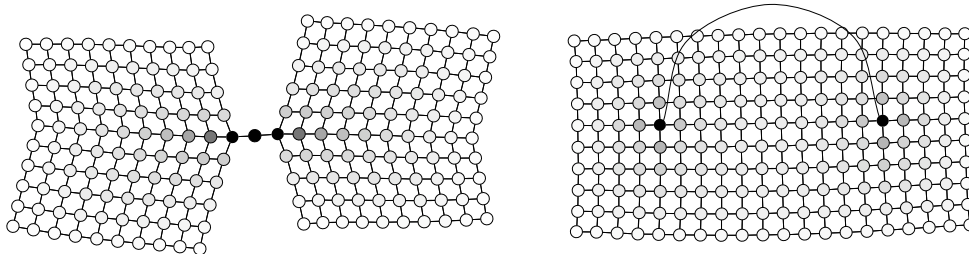

Figure 1: A visual representation of betweenness on two sample graphs.

## 2  Skill Definition

We assume that the agent's interaction with its environment may be represented as a Markov Decision Process (MDP). The *interaction graph* is a directed graph in which the vertices represent the states of the MDP and the edges represent possible state transitions brought about by available actions. Specifically, the edge $u \rightarrow v$ is present in the graph if and only if the corresponding state transition has a strictly positive probability through the execution of at least one action. The weight on each edge is the expected cost of the transition, or expected negative reward.

Our claim is that states that have a pivotal role in efficiently navigating the interaction graph are useful subgoals to reach and that a useful measure for evaluating how pivotal a vertex $v$ is

$$\sum_{s \neq t \neq v} \frac{\sigma_{st}(v)}{\sigma_{st}} w_{st},$$

where $\sigma_{st}$ is the number of shortest paths from vertex $s$ to vertex $t$, $\sigma_{st}(v)$ is the number of such paths that pass through vertex $v$, and $w_{st}$ is the weight assigned to paths from vertex $s$ to vertex $t$.

With uniform path weights, the above expression equals *betweenness*, a measure of centrality on graphs [1, 2]. It gives the fraction of shortest paths on the graph (between all possible sources and destinations) that pass through the vertex of interest. If there are multiple shortest paths from a given source to a given destination, they are given equal weights that sum to one. Betweenness may be computed in $O(nm)$ time and $O(n + m)$ space on unweighted graphs with $n$ nodes and $m$ edges [10]. On weighted graphs, the space requirement remains the same, but the time requirement increases to $O(nm + n^2 logn)$.

In our use of betweenness, we include path weights to take into account the reward function. Depending on the reward function—or a probability distribution over possible reward functions—some parts of the interaction graph may be given more weight than others, depending on how well they serve the agent's needs.

We define as subgoals those states that correspond to local maxima of betweenness on the interaction graph, in other words, states that have a higher betweenness than other states in their neighborhood. Here, we use a simple definition of *neighborhood*, including in it only the states that are one hop away, which may be revised in the future. Skills for efficiently reaching the local maxima of betweenness represent a set of behaviors that may be combined in different ways to efficiently reach different regions, serving as useful building blocks for navigating the graph.

Figure 1 is a visual representation of betweenness on two sample graphs, computed using uniform edge and path weights. The gray-scale shading on the vertices corresponds to the relative values of betweenness, with black representing the highest betweenness on the graph and white representing the lowest. The graph on the left corresponds to a gridworld in which a doorway connects two rooms. The graph on the right has a doorway of a different type: an edge connecting two otherwise distant nodes. In both graphs, states that are local maxima of betweenness correspond to our intuitive choice of subgoals.

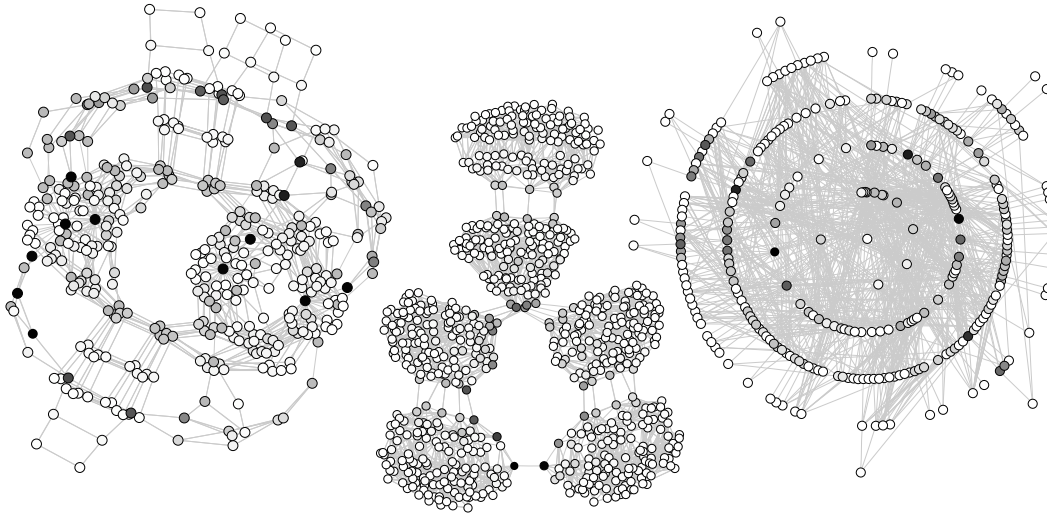

Figure 2: Betweenness in Taxi, Playroom, and Tic-Tac-Toe (from left to right). Edge directions are omitted in the figure.

## 3    Examples

We appled the skill definition of Section 2 to various domains in the literature: Taxi [11], Play-room [12, 13], and the game of Tic-Tac-Toe. Interaction graphs of these domains, displaying be-tweenness values as gray-scale shading on the vertices, are shown in Figure 2. In Taxi and Playroom, graph layouts were determined by a force-directed algorithm that models the edges as springs and minimizes the total force on the system. We considered a node to be a local maximum if its be-tweenness was higher than or equal to those of its immediate neighbors, taking into account both incoming and outgoing edges. Unless stated otherwise, actions had uniform cost and betweenness was computed using uniform path weights.

**Taxi**  This domain includes a taxi and a passenger on the $5 \times 5$ grid shown in Figure 4. At each grid location, the taxi has six primitive actions: `north`, `east`, `south`, `west`, `pick-up`, and `put-down`. The navigation actions succeed in moving the taxi in the intended direction with prob-ability 0.8; with probability 0.2, the action takes the taxi to the right or left of the intended direction. If the direction of movement is blocked, the taxi remains in the same location. `Pick-up` places the passenger in the taxi if the taxi is at the passenger location; otherwise it has no effect. Similarly, `put-down` delivers the passenger if the passenger is inside the taxi and the taxi is at the destina-tion; otherwise it has no effect. The source and destination of all passengers are chosen uniformly at random from among the grid squares R, G, B, Y. We used a continuing version of this problem in which a new passenger appears after each successful delivery.

The highest local maxima of betweenness are at the four regions of the graph that correspond to passenger delivery. Other local maxima belong to one of the following categories: (1) taxi is at the passenger location[1], (2) taxi is at one of the passenger wait locations with the passenger in the taxi[2], (3) taxi and passenger are both at destination, (4) the taxi is at $x = 2$, $y = 3$, a navigational bottleneck on the grid, and (5) the taxi is at $x = 3$, $y = 3$, another navigational bottleneck. The corresponding skills are (approximately) those that take the taxi to the passenger location, to the destination (having picked up the passenger), or to a navigational bottleneck. These skills closely resemble those that are hand-coded for this domain in the literature.

**Playroom**  We created a Markov version of this domain in which an agent interacts with a number of objects in its surroundings: a light switch, a ball, a bell, a button for turning music on and off,

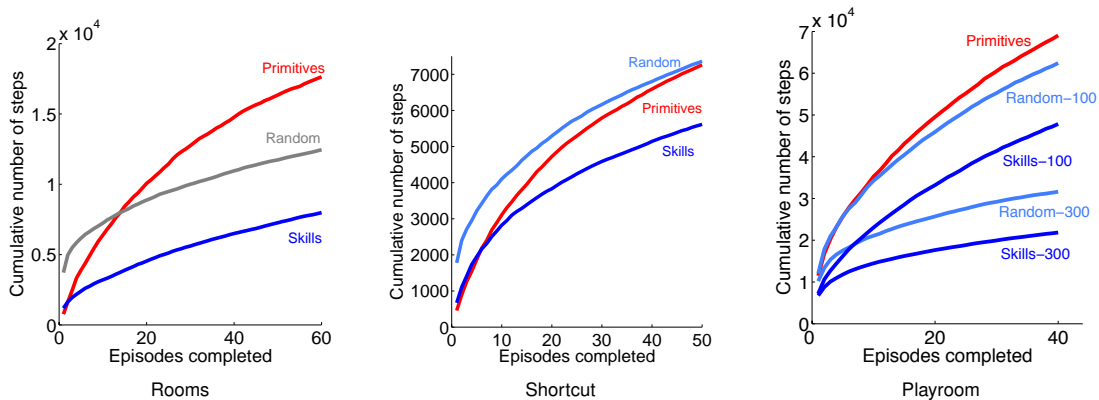

Figure 3: Learning performance in Rooms, Shortcut, and Playroom.

and a toy monkey. The agent has an eye, a hand, and a marker it can place on objects. Its actions consist of looking at a randomly selected object, looking at the object in its hand, holding the object it is looking at, looking at the object that the marker is placed on, placing the marker on the object it is looking at, moving the object in its hand to the location it is looking at, flipping the light switch, pressing the music button, and hitting the ball towards the marker. The first two actions succeed with probability 1, while the remaining actions succeed with probability 0.75, producing no change in the environment if they fail. In order to operate on an object, the agent must be looking at the object and holding the object in its hand. To be able to press the music button successfully, the light should be on. The toy monkey starts to make frightened sounds if the bell is rung while the music is playing; it stops only when the music is turned off. If the ball hits the bell, the bell rings for one decision stage.

The MDP state consists of the object that the agent is looking at, the object that the agent is holding, the object that the marker is placed on, music (on/off), light (on/off), monkey (frightened/not), and bell (ringing/not). The six different clusters of the interaction graph in Figure 2 emerge naturally from the force-directed layout algorithm and correspond to the different settings of the music, light, and monkey variables. There are only six such clusters because not all variable combinations are possible. Betweenness peaks at regions that immediately connect neighboring clusters, corresponding to skills that change the setting of the music, light, or monkey variables.

**Tic-Tac-Toe** In the interaction graph, the node at the center of the interaction graph is the empty board, with other board configurations forming rings around it with respect to their distance from this initial configuration. The innermost ring shows states in which both players have played a single turn. The agent played first. The opponent followed a policy that (1) placed the third mark in a row, whenever possible, winning the game, (2) blocked the agent from completing a row, and (3) placed its mark on a random empty square, with decreasing priority. Our state representation was invariant with respect to rotational and reflective symmetries of the board. We assigned a weight of +1 to paths that terminate at a win for the agent and 0 to all other paths. The state with the highest betweenness is the one shown in Figure 4. The agent is the X player and will go next. This state gives the agent two possibilities for setting up a fork (board locations marked with *), creating an opportunity to win on the next turn. There were nine other local maxima that similarly allowed the agent to immediately create a fork. In addition, there were a number of "trivial" local maxima that allowed the agent to immediately win the game.

## 4 Empirical Performance

We evaluated the impact of our skills on the agent's learning performance in Taxi, Playroom, Tic-Tac-Toe, and two additional domains, called Rooms and Shortcut, whose interaction graphs are those presented in Figure 1. Rooms is a gridworld in which a doorway connects two rooms. At each state, the available actions are `north`, `south`, `east`, and `west`. They move the agent in the intended direction with probability 0.8 and in a uniform random direction with probability 0.2. The local

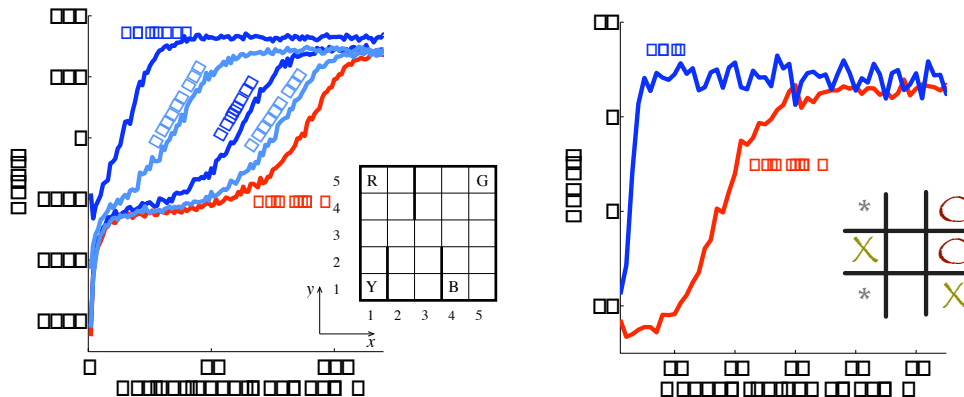

Figure 4: Learning performance in Taxi and Tic-Tac-Toe.

maxima of betweenness are the two states that surround the doorway, which have a slightly higher betweenness than the doorway itself. The transition dynamics of Shortcut is identical, except there is one additional long-range action, connecting two particular states, which are the local maxima of betweenness in this domain.

We represented skills using the options framework [14, 15]. The initiation set was restricted to include a certain number of states and included those states with the least distance to the subgoal on the interaction graph. The skills terminated with probability one outside the initiation set and at the subgoal, with probability zero at all other states. The skill policy was the optimal policy for reaching the subgoal. We compared three agents: one that used only the primitive actions of the domain, one that used primitives and our skills, and one that used primitives and a control group of skills whose subgoals were selected randomly. The number of subgoals used and the size of the initiation sets were identical in the two skill conditions. The agent used Q-learning with $\epsilon$-greedy exploration with $\epsilon = 0.05$. When using skills, it performed both intra-option and macro-Q updates [16]. The learning rate ($\alpha$) was kept constant at 0.1. Initial Q-values were 0. Discount rate $\gamma$ was set to 1 in episodic tasks, to 0.99 in continuing tasks.

Figure 3 shows performance results in Rooms, Shortcut, and PlayRoom, where we had the agent perform 100 different episodic tasks, choosing a single goal state uniformly randomly in each task. The reward was $-0.001$ for each transition and an additional $+1$ for transitions into the goal state. The initial state was selected randomly. The labels in the figure indicate the size of the initiation sets. If no number is present, the skills were made available everywhere in the domain. The availability of our skills—those that were identified using local maxima of betweenness—revealed a big improvement compared to using primitive actions only. In some cases, random skills improved performance as well, but this improvement was much smaller than that obtained by our skills.

Figure 4 shows similar results in Taxi and Tic-Tac-Toe. The figure shows mean performance in 100 trials. In Taxi, we examined performance on the single continuing task that rewarded the agent for delivering passengers. Reward was $-1$ for each action, an additional $+50$ for passenger delivery, and an additional $-10$ for an unsuccessful `pick-up` or `put-down`. In Tic-Tac-Toe, the agent received a reward of $-0.001$ for each action, an additional $+1$ for winning the game, and an additional $-1$ for losing. Creating an individual skill for reaching each of the identified subgoals (which is what we have done in other domains) generates skills that are not of much use in Tic-Tac-Toe because reaching any particular board configuration is usually not possible. Instead, we defined a single skill with multiple subgoals—the ten local maxima of betweenness that allow the agent to setup a fork. We set the initial Q-value of this skill to 1 at the start state to ensure that the skill got executed frequently enough. It is not clear what this single skill can be meaningfully compared to, so we do not provide a control condition with randomly-selected subgoals.

Our analysis shows that, in a diverse set of domains, the skill definition of Section 2 gives rise to skills that are consistent with common sense, are similar to skills people handcraft for these domains, and improve learning performance. The improvements in performance are greater than those observed when using a control group of randomly-generated skills, suggesting that they should not be attributed to the presence of skills alone but to the presence of the specific skills that are formed based on betweenness.

## 5    Related Work

A graphical approach to forming high-level behavioral units was first suggested by Amarel in his classic analysis of the missionaries and cannibals problem [17]. Amarel advocated representing action consequences in the environment as a graph and forming skills that correspond to navigating this graph by exploiting its structural regularities. He did not, however, propose any general mechanism that can be used for this purpose.

Our skill definition captures the "bottleneck" concept, which has inspired many of the existing skill discovery algorithms [3, 6, 4, 5, 7, 8, 9]. There is clearly an overlap between our skills and the skills that are generated by these algorithms. Here, we review these algorithms, with a focus on the extent of this overlap and sample efficiency.

McGovern & Barto [3] examine past trajectories to identify states that are common in successful trajectories but not in unsuccessful ones. An important concern with their method is its need for excessive exploration of the environment. It can be applied only after the agent has successfully performed the task at least once. Typically, it requires many additional successful trajectories. Furthermore, a fundamental property of this algorithm prevents it from identifying a large portion of our subgoals. It examines different paths that reach the same destination, while we look for the most efficient ways of navigating between different source and destination pairs. Bottlenecks that are not on the path to the goal state would not be identified by this algorithm, while we consider such states to be useful subgoals.

Stolle & Precup [4] and Stolle [5] address this last concern by obtaining their trajectories from multiple tasks that start and terminate at different states. As the number of tasks increases, the subgoals identified by their algorithms become more similar to ours. Unfortunately, however, sample efficiency is even a larger concern with these algorithms, because they require the agent to have already identified the optimal policy—not for only a single task, but for many different tasks in the domain.

Menache et al. [6] and Mannor et al. [8] take a graphical approach and use the MDP state-transition graph to identify subgoals. They apply a clustering algorithm to partition the graph into blocks and create skills that efficiently take the agent to states that connect the different blocks. The objective is to identify blocks that are highly connected within themselves but weakly connected to each other. Different clustering techniques and cut metrics may be used towards this end. Rooms and Playroom are examples of where these algorithms can succeed. Tic-Tac-Toe and Shortcut are examples of where they fail.

Şimşek, Wolfe & Barto [9] address certain shortcomings of global graph partitioning by constructing their graphs from short trajectories. Şimşek & Barto [7] take a different approach and search for states that introduce short-term novelty. Although their algorithm does not explicitly use the connectivity structure of the domain, it shares some of the limitations of graph partitioning as we discuss more fully in the next section. We claim that the more fundamental property that makes a doorway a useful subgoal is that it is *between* many source-destination pairs and that graph partitioning can not directly tap into this property, although it can sometimes do it indirectly.

## 6    An Incremental Discovery Algorithm

Our skill definition may be used directly to form a set of skills suitable for a given environment. Because of its reliance on complete knowledge of the interaction graph and the computational cost of betweenness, the use of our approach as a skill-discovery method is limited, although there are conditions under which it would be useful. An important research question is whether approximate methods may be developed that do not require complete representation of the interaction graph.

Although betweenness of a given vertex is a global graph property that can not be estimated reliably without knowledge of the entire graph, it should be possible to reliably determine the local maxima of betweenness using limited information. Here, we investigate this possibility by combining the descriptive contributions of the present paper with algorithmic insights of earlier work. In particular, we apply the statistical approach from Şimşek & Barto [7] and Şimşek, Wolfe & Barto [9] using the skill description in the present paper.

The resulting algorithm is founded on the premise that local maxima of betweenness of the interaction graph are likely to be local maxima on its subgraphs. While any single subgraph would not be particularly useful to identify such vertices, a collection of subgraphs may allow us to correctly identify them. The algorithm proceeds as follows. The agent uses short trajectories to construct subgraphs of the interaction graph and identifies the local maxima of betweenness on these subgraphs. From each subgraph, it obtains a new observation for every state represented on it. This is a positive observation if the state is a local maximum, a negative observation otherwise. We use the decision rule from Şimşek, Wolfe & Barto [9], making a particular state a subgoal if there are at least $n_o$ observations on this state and if the proportion of positive observations is at least $p_+$. The agent continues this incremental process indefinitely.

Figure 5 shows the results of applying this algorithm on two domains. The first is a gridworld with six rooms. The second is also a gridworld, but the grid squares are one of two types with different rewards. The lightly colored squares produce a reward of $-0.001$ for actions that originate on them, while the darker squares produce $-0.1$. The reward structure creates two local maxima of betweenness on the graph. These are the regions that look like doorways in the figure—they are useful subgoals for the same reasons that doorways are. Graph partitioning does not succeed in identifying these states because the structure is not created through node connectivity. Similarly, the algorithms by Şimşek & Barto [7] and Şimşek, Wolfe & Barto [9] are also not suitable for this domain. We applied them and found that they identified very few subgoals ($<0.05$/trial) randomly distributed in the domain.

In both domains, we had the agent perform a random walk of 40,000 steps. Every 1000 transitions, the agent created a new interaction graph using the last 1000 transitions. Figure 5 shows the number of times each state was identified as a subgoal in 100 trials, using $n_o = 10$, $p_+ = 0.2$. The individual graphs had on average 156 nodes in the six-room gridworld and 224 nodes in the other one.

We present this algorithm here as a proof of concept, to demonstrate the feasibility of incremental algorithms. An interesting direction is to develop algorithms that actively explore to discover local maxima of betweenness rather than only passively mining available trajectories.

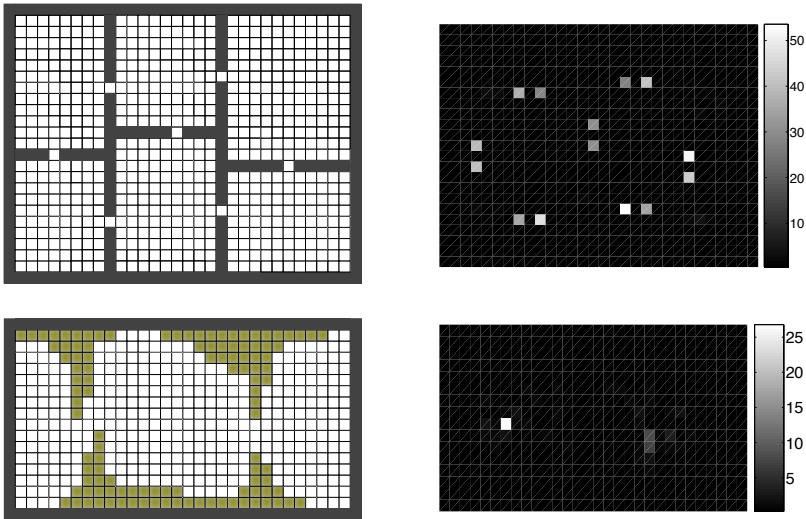

Figure 5: Subgoal frequency in 100 trials using the incremental discovery algorithm.

**Acknowledgments**

This work is supported in part by the National Science Foundation under grant CNS-0619337 and by the Air Force Office of Scientific Research under grant FA9550-08-1-0418. Any opinions, findings, conclusions or recommendations expressed here are those of the authors and do not necessarily reflect the views of the sponsors.

## Footnotes

[1]Except when passenger is waiting at Y, in which case, the taxi is at $x = 1$, $y = 3$.

[2]For wait location Y, the corresponding subgoal has the taxi at $x = 1$, $y = 3$, having picked up the passenger.

# References

[1] L. C. Freeman. A set of measures of centrality based upon betweenness. *Sociometry*, 40:35–41, 1977.

[2] L. C. Freeman. Centrality in social networks: Conceptual clarification. *Social Networks*, 1:215–239, 1979.

[3] A. McGovern and A. G. Barto. Automatic discovery of subgoals in reinforcement learning using diverse density. In *Proceedings of the Eighteenth International Conference on Machine Learning*, 2001.

[4] M. Stolle and D. Precup. Learning options in reinforcement learning. *Lecture Notes in Computer Science*, 2371:212–223, 2002.

[5] M. Stolle. Automated discovery of options in reinforcement learning. Master's thesis, McGill University, 2004.

[6] I. Menache, S. Mannor, and N. Shimkin. Q-Cut - Dynamic discovery of sub-goals in reinforcement learning. In *Proceedings of the Thirteenth European Conference on Machine Learning*, 2002.

[7] Ö. Şimşek and A. G. Barto. Using relative novelty to identify useful temporal abstractions in reinforcement learning. In *Proceedings of the Twenty-First International Conference on Machine Learning*, 2004.

[8] S. Mannor, I. Menache, A. Hoze, and U. Klein. Dynamic abstraction in reinforcement learning via clustering. In *Proceedings of the Twenty-First International Conference on Machine Learning*, 2004.

[9] Ö. Şimşek, A. P. Wolfe, and A. G. Barto. Identifying useful subgoals in reinforcement learning by local graph partitioning. In *Proceedings of the Twenty-Second International Conference on Machine Learning*, 2005.

[10] U. Brandes. A faster algorithm for betweenness centrality. *Journal of Mathematical Sociology*, 25(2):163–177, 2001.

[11] T. G. Dietterich. Hierarchical reinforcement learning with the MAXQ value function decomposition. *Journal of Artificial Intelligence Research*, 13:227–303, 2000.

[12] A. G. Barto, S. Singh, and N. Chentanez. Intrinsically motivated learning of hierarchical collections of skills. In *Proceedings of the Third International Conference on Developmental Learning*, 2004.

[13] S. Singh, A. G. Barto, and N. Chentanez. Intrinsically motivated reinforcement learning. In *Advances in Neural Information Processing Systems*, 2005.

[14] R. S. Sutton, D. Precup, and S. P. Singh. Between MDPs and Semi-MDPs: A framework for temporal abstraction in reinforcement learning. *Artificial Intelligence*, 112(1-2):181–211, 1999.

[15] D. Precup. *Temporal abstraction in reinforcement learning*. PhD thesis, University of Massachusetts Amherst, 2000.

[16] A. McGovern, R. S. Sutton, and A. H. Fagg. Roles of macro-actions in accelerating reinforcement learning. In *Grace Hopper Celebration of Women in Computing*, 1997.

[17] S. Amarel. On representations of problems of reasoning about actions. In *Machine Intelligence 3*, pages 131–171. Edinburgh University Press, 1968.

